# Learning Factored Representations for Partially Observable Markov Decision Processes

**Brian Sallans**

Department of Computer Science
University of Toronto
Toronto M5S 2Z9 Canada

Gatsby Computational Neuroscience Unit*
University College London
London WC1N 3AR U.K.

*sallans@cs.toronto.edu*

## Abstract

The problem of reinforcement learning in a non-Markov environment is explored using a dynamic Bayesian network, where conditional independence assumptions between random variables are compactly represented by network parameters. The parameters are learned on-line, and approximations are used to perform inference and to compute the optimal value function. The relative effects of inference and value function approximations on the quality of the final policy are investigated, by learning to solve a moderately difficult driving task. The two value function approximations, linear and quadratic, were found to perform similarly, but the quadratic model was more sensitive to initialization. Both performed below the level of human performance on the task. The dynamic Bayesian network performed comparably to a model using a localist hidden state representation, while requiring exponentially fewer parameters.

## 1 Introduction

Reinforcement learning (RL) addresses the problem of learning to act so as to maximize a reward signal provided by the environment. Online RL algorithms try to find a policy which maximizes the expected time-discounted reward. They do this through experience by performing sample backups to learn a value function over states or state-action pairs.

If the decision problem is Markov in the observable states, then the optimal value function over state-action pairs yields all of the information required to find the optimal policy for the decision problem. When complete knowledge of the environment is not available, states which are different may look the same; this uncertainty is called *perceptual aliasing* [1], and causes decision problems to have dynamics which are non-Markov in the perceived state.

## 1.1 Partially observable Markov decision processes

Many interesting decision problems are not Markov in the inputs. A partially observable Markov decision process (POMDP) is a formalism in which it is assumed that a process is Markov, but with respect to some unobserved (i.e. "hidden") random variable. The state of the variable at time $t$, denoted $s^t$, is dependent only on the state at the previous time step and on the action performed. The currently-observed evidence is assumed to be independent of previous states and observations given the current state.

The state of the hidden variable is not known with certainty, so a belief state is maintained instead. At each time step, the beliefs are updated by using Bayes' theorem to combine the belief state at the previous time step (passed through a model of the system dynamics) with newly observed evidence. In the case of discrete time and finite discrete state and actions, a POMDP is typically represented by conditional probability tables (CPTs) specifying emission probabilities for each state, and transition probabilities and expected rewards for states and actions. This corresponds to a hidden Markov model (HMM) with a distinct transition matrix for each action. The hidden state is represented by a single random variable that can take on one of $K$ values. Exact belief updates can be computed using Bayes' rule.

The value function is not over the discrete state, but over the real-valued belief state. It has been shown that the value function is piecewise linear and convex [2]. In the worst case, the number of linear pieces grows exponentially with the problem horizon, making exact computation of the optimal value function intractable.

Notice that the localist representation, in which the state is encoded in a single random variable, is exponentially inefficient: Encoding $n$ bits of information about the state of the process requires $2^n$ possible hidden states. This does not bode well for the abilities of models which use this representation to scale up to problems with high-dimensional inputs and complex non-Markov structure.

## 1.2 Factored representations

A Bayesian network can compactly represent the state of the system in a set of random variables [3]. A two time-slice dynamic Bayesian network (DBN) represents the system at two time steps [4]. The conditional dependencies between random variables from time $t$ to time $t + 1$, and within time step $t$, are represented by edges in a directed acyclic graph. The conditional probabilities can be stored explicitly, or parameterized by weights on edges in the graph.

If the network is densely-connected then inference is intractable [5]. Approximate inference methods include Markov chain Monte Carlo [6], variational methods [7], and belief state simplification [8].

In applying a DBN to a large problem there are three distinct issues to disentangle: How well does a parameterized DBN capture the underlying POMDP; how much is the DBN hurt by approximate inference; and how good must the approximation of the value function be to achieve reasonable performance? We try to tease these issues apart by looking at the performance of a DBN on a problem with a moderately large state-space and non-Markov structure.

## 2 The algorithm

We use a fully-connected dynamic sigmoid belief network (DSBN) [9], with $K$ units at each time slice (see figure 1). The random variables $s_i$ are binary, and conditional proba-

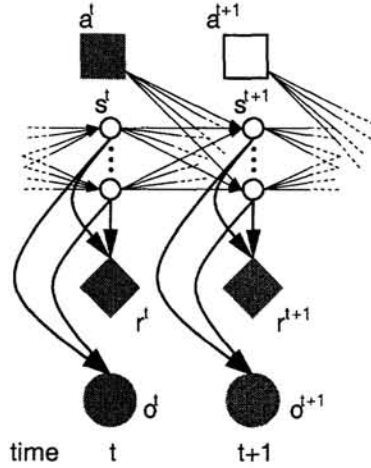

Figure 1: Architecture of the dynamic sigmoid belief network. Circles indicate random variables, where a filled circle is observed and an empty circle is unobserved. Squares are action nodes, and diamonds are rewards.

bilities relating variables at adjacent time-steps are encoded in action-specific weights:

$$P(s_i^{t+1} = 1|\{s_k^t\}_{k=1}^K, a^t) = \sigma\left(\sum_{k=1}^K w_{ik}^{a^t} s_k^t\right) \tag{1}$$

where $w_{ik}^{a^t}$ is the weight from the $i^{th}$ unit at time step $t$ to the $k^{th}$ unit at time step $t + 1$, assuming action $a^t$ is taken at time $t$. The nonlinearity is the usual sigmoid function: $\sigma(x) = 1/1 + \exp\{-x\}$. Note that a bias can be incorporated into the weights by clamping one of the binary units to 1.

The observed variables are assumed to be discrete; the conditional distribution of an output given the hidden state is multinomial and parameterized by output weights. The probability of observing an output with value $l$ is given by:

$$P(o^t = l|\{s_k^t\}_{k=1}^K) = \frac{\exp\left\{\sum_{k=1}^K u_{kl} s_k^t\right\}}{\sum_{m=1}^{|O|} \exp\left\{\sum_{k=1}^K u_{km} s_k^t\right\}} \tag{2}$$

where $o^t \in O$ and $u_{kl}$ denotes the output weight from hidden unit $k$ to output value $l$.

## 2.1 Approximate inference

Inference in the fully-connected Bayesian network is intractable. Instead we use a variational method with a fully-factored approximating distribution:

$$P(s^t|s^{t-1}, a^{t-1}, o^t) \overset{\triangle}{=} P_{s^t} \approx \prod_{k=1}^K \mu_k^{s_k^t}(1 - \mu_k)^{1-s_k^t} \tag{3}$$

where the $\mu_k$ are variational parameters to be optimized. This is the standard mean-field approximation for a sigmoid belief network [10]. The parameters $\mu$ are optimized by iterating the mean-field equations, and converge in a few iterations. The values of the variational parameters at time $t$ are held fixed while computing the values for step $t + 1$. This is analogous to running only the forward portion of the HMM forward-backward algorithm [11].

The parameters of the DSBN are optimized online using stochastic gradient ascent in the log-likelihood:

$$\mathbf{W}^{a^t} \leftarrow \mathbf{W}^{a^t} + \alpha_W \left[\mu^{t+1} - \sigma\left(\mathbf{W}^{a^t}\mu^t\right)\right] \cdot \mu^{t\top}$$

$$\mathbf{U} \leftarrow \mathbf{U} + \alpha_U \left[\nu^t - \frac{\exp\{\mathbf{U}\mu\}}{\sum_k \exp\{[\mathbf{U}\mu]_k\}}\right] \cdot \mu^{t\top} \tag{4}$$

where $\mathbf{W}$ and $\mathbf{U}$ are the transition and emission matrices respectively, $\alpha_W$ and $\alpha_U$ are learning rates, the vector $\mu$ contains the fully-factored approximate belief state, and $\nu$ is a vector of zeros with a one in the $o^{t^{th}}$ place. The notation $[\cdot]_k$ denotes the $k^{th}$ element of a vector (or $k^{th}$ column of a matrix).

## 2.2  Approximating the value function

Computing the optimal value function is also intractable. If a factored state-space representation is appropriate, it is natural (if extreme) to assume that the state-action value function can be decomposed in the same way:

$$Q(P_{\mathbf{s}^t}, a^t) \approx \sum_{k=1}^{K} Q_k(\mu_k^t, a^t) \overset{\triangle}{=} Q_F(\mu, a^t) \tag{5}$$

This simplifying assumption is still not enough to make finding the optimal value function tractable. Even if the states were completely independent, each $Q_k$ would still be piecewise-linear and convex, with the number of pieces scaling exponentially with the horizon. We test two approximate value functions, a linear approximation:

$$Q_F(\mu^t, a^t) = \sum_{k=1}^{K} q_{k,a^t}\, \mu_k = [\mathbf{Q}]_{a^t}{}^{\top} \cdot \mu \tag{6}$$

and a quadratic approximation:

$$\begin{aligned} Q_F(\mu^t, a^t) &= \sum_{k=1}^{K} \phi_{k,a^t}\, \mu_k^2 + q_{k,a^t}\, \mu_k + b_{a^t} \\ &= [\mathbf{\Phi}]_{a^t}{}^{\top} \cdot (\mu \odot \mu) + [\mathbf{Q}]_{a^t}{}^{\top} \cdot \mu + [\mathbf{b}]_{a^t} \end{aligned} \tag{7}$$

Where $\mathbf{\Phi}$, $\mathbf{Q}$ and $\mathbf{b}$ are parameters of the approximations. The notation $[\cdot]_i$ denotes the $i^{th}$ column of a matrix, $[\cdot]^{\top}$ denotes matrix transpose and $\odot$ denotes element-wise vector multiplication.

We update each term of the factored approximation with a modified Q-learning rule [12], which corresponds to a delta-rule where the target for input $\mu$ is $r^t + \gamma \max_a Q_F(\mu^{t+1}, a)$:

$$\begin{aligned} \phi_{k,a^t} &\leftarrow \phi_{k,a^t} + \alpha\, \mu_k^2\, E_B \\ q_{k,a^t} &\leftarrow q_{k,a^t} + \alpha\, \mu_k\, E_B \\ b_{a^t} &\leftarrow b_{a^t} + \alpha\, E_B \end{aligned} \tag{8}$$

Here $\alpha$ is a learning rate, $\gamma$ is the temporal discount factor, and $E_B$ is the Bellman residual:

$$E_B = r^t + \gamma \max_a Q_F(\mu^{t+1}, a) - Q_F(\mu^t, a^t) \tag{9}$$

## 3  Experimental results

The "New York Driving" task [13] involves navigating through slower and faster one-way traffic on a multi-lane highway. The speed of the agent is fixed, and it must change lanes to avoid slower cars and move out of the way of faster cars. If the agent remains in front of a faster car, the driver of the fast car will honk its horn, resulting in a reward of $-1.0$. Instead of colliding with a slower car, the agent can squeeze past in the same lane, resulting in a reward of $-10.0$. A time step with no horns or lane-squeezes constitutes clear progress, and is rewarded with $+0.1$. See [13] for a detailed description of this task.

Table 1: Sensory input for the New York driving task

| Dimension | Size | Values |
|---|---|---|
| Hear horn | 2 | yes, no |
| Gaze object | 3 | truck, shoulder, road |
| Gaze speed | 2 | looming, receding |
| Gaze distance | 3 | far, near, nose |
| Gaze refined distance | 2 | far-half, near-half |
| Gaze colour | 6 | red, blue, yellow, white, gray, tan |

A modified version of the New York Driving task was used to test our algorithm. The task was essentially the same as described in [13], except that the "gaze side" and "gaze direction" inputs were removed. See table 1 for a list of the modified sensory inputs.

The performance of a number of algorithms and approximations were measured on the task: a random policy; Q-learning on the sensory inputs; a model with a localist representation (i.e. the hidden state consisted of a single multinomial random variable) with linear and quadratic approximate value functions; the DSBN with mean-field inference and linear and quadratic approximations; and a human driver. The localist representation used the linear Q-learning approximation of [14], and the corresponding quadratic approximation. The quadratic approximations were trained both from random initialization, and from initialization with the corresponding learned linear models (and random quadratic portion). The non-human algorithms were each trained for 100000 iterations, and in each case a constant learning rate of 0.01 and temporal decay rate of 0.9 were used. The human driver (the author) was trained for 1000 iterations using a simple character-based graphical display, with each iteration lasting 0.5 seconds.

Stochastic policies were used for all RL algorithms, with actions being chosen from a Boltzmann distribution with temperature decreasing over time:

$$P(a^t|\mu^t) = \frac{1}{Z_B} \exp\{Q_F(\mu^t, a^t)/T\} \tag{10}$$

The DSBN had 4 hidden units per time slice, and the localist model used a multinomial with 16 states. The Q-learner had a table representation with 2160 entries. After training, each non-human algorithm was tested for 20 trials of 5000 time steps each. The human was tested for 2000 time steps, and the results were renormalized for comparison with the other methods. The results are shown in figure 2. All results were negative, so lower numbers indicate better performance in the graph. The error bars show one standard deviation across the 20 trials.

There was little performance difference between the localist representation and the DSBN but, as expected, the DSBN was exponentially more efficient in its hidden-state representation. The linear and quadratic approximations performed comparably, but well below human performance. However, the DSBN with quadratic approximation was more sensitive to initialization. When initialized with random parameter settings, it failed to find a good policy. However, it did converge to a reasonable policy when the linear portion of the quadratic model was initialized with a previously learned linear model.

The hidden units in the DSBN encode useful features of the input, such as whether a car was at the "near" or "nose" position. They also encode some history, such as current gaze direction. This has advantages over a simple stochastic policy learned via Q-learning: If the Q-learner knows that there is an oncoming car, it can randomly select to look left or right. The DSBN systematically looks to the left, and then to the right, wasting fewer actions.

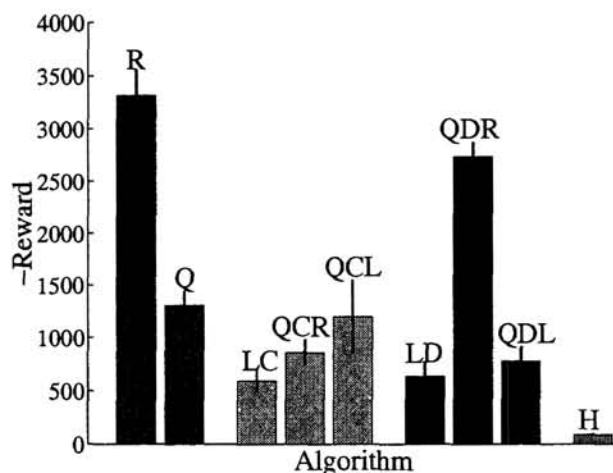

Figure 2: Results on the New York Driving task for nine algorithms: R=random; Q=Q-learning; LC=linear multinomial; QCR=quadratic multinomial, random init.; QCL=quadratic multinomial, linear init; LD=linear DSBN; QDR=quadratic DSBN, random init.; QDL=quadratic DSBN, linear init.; H=human

## 4  Discussion

The DSBN performed better than a standard Q-learner, and comparably to a model with a localist representation, despite using approximate inference and exponentially fewer parameters. This is encouraging, since an efficient encoding of the state is a prerequisite for tackling larger decision problems. Less encouraging was the value-function approximation: When compared to human performance, it is clear that all methods are far from optimal, although again the factored approximation of the DSBN did not hurt performance relative to the localist multinomial representation. The sensitivity to initialization of the quadratic approximation is worrisome, but the success of initializing from a simpler model suggests that staged learning may be appropriate, where simple models are learned and used to initialize more complex models. These findings echo those of [14] in the context of learning a non-factored approximate value function.

There are a number of related works, both in the fields of reinforcement learning and Bayesian networks. We use the sigmoid belief network mean-field approximation given in [10], and discussed in the context of time-series models (the "fully factored" approximation) in [15]. Approximate inference in dynamic Bayesian networks has been discussed in [15] and [8]. The additive factored value function was used in the context of factored MDPs (with no hidden state) in [16], and the linear Q-learning approximation was given in [14]. Approximate inference was combined with more sophisticated value function approximation in [17]. To our knowledge, this is the first attempt to explore the practicality of combining all of these techniques in order to solve a single problem.

There are several possible extensions. As described above, the representation learned by the DSBN is not tuned to the task at hand. The reinforcement information could be used to guide the learning of the DSBN parameters[18, 13]. Also, if this were done, then the reinforcement signals would provide additional evidence as to what state the POMDP is in, and could be used to aid inference. More sophisticated function approximation could be used [17]. Finally, although this method appears to work in practice, there is no guarantee that the reinforcement learning will converge. We view this work as an encouraging first step, with much further study required.

## 5  Conclusions

We have shown that a dynamic Bayesian network can be used to construct a compact representation useful for solving a decision problem with hidden state. The parameters of the DBN can be learned from experience. Learning occurs despite the use of simple value-

function approximations and mean-field inference. Approximations of the value function result in good performance, but are clearly far from optimal. The fully-factored assumptions made for the belief state and the value function do not appear to impact performance, as compared to the non-factored model. The algorithm as presented runs entirely on-line by performing "forward" inference only. There is much room for future work, including improving the utility of the factored representation learned, and the quality of approximate inference and the value function approximation.

**Acknowledgments**

We thank Geoffrey Hinton, Zoubin Ghahramani and Andy Brown for helpful discussions, the anonymous referees for valuable comments and criticism, and particularly Peter Dayan for helpful discussions and comments on an early draft of this paper. This research was funded by NSERC Canada and the Gatsby Charitable Foundation.

## Footnotes

*Correspondence address

# References

[1] S.D. Whitehead and D.H. Ballard. Learning to perceive and act by trial and error. *Machine Learning*, 7, 1991.

[2] E.J. Sondik. The optimal control of partially observable Markov processes over the infinite horizon: Discounted costs. *Operations Research*, 26:282–304, 1973.

[3] J. Pearl. *Probabilistic Reasoning in Intelligent Systems: Networks of Plausible Inference*. Morgan Kaufmann, San Mateo, CA, 1988.

[4] T. Dean and K. Kanazawa. A model for reasoning about persistence and causation. *Computational Intelligence*, 5, 1989.

[5] Gregory F. Cooper. The computational complexity of probabilistic inference using Bayesian belief networks. *Artificial Intelligence*, 42:393–405, 1990.

[6] R. M. Neal. Probabilistic inference using Markov chain Monte Carlo methods. Technical Report CRG-TR-93-1, Department of Computer Science, University of Toronto, 1993.

[7] M.I. Jordan, Z. Ghahramani, T.S. Jaakkola, and L.K. Saul. An introduction to variational methods for graphical models. *Machine Learning*, 1999. in press.

[8] X. Boyen and D. Koller. Tractable inference for complex stochastic processes. In *Proc. UAI'98*, 1998.

[9] R. M. Neal. Connectionist learning of belief networks. *Artificial Intelligence*, 56:71–113, 1992.

[10] L. K. Saul, T. Jaakkola, and M. I. Jordan. Mean field theory for sigmoid belief networks. *Journal of Artificial Intelligence Research*, 4:61–76, 1996.

[11] Lawrence R. Rabiner and Biing-Hwang Juang. An introduction to hidden Markov models. *IEEE ASSAP Magazine*, 3:4–16, January 1986.

[12] C.J.C.H. Watkins and P. Dayan. Q-learning. *Machine Learning*, 8:279–292, 1992.

[13] A.K. McCallum. *Reinforcement learning with selective perception and hidden state*. Dept. of Computer Science, Universiy of Rochester, Rochester NY, 1995. Ph.D. thesis.

[14] M.L. Littman, A.R. Cassandra, and L.P. Kaelbling. Learning policies for partially observable environments: Scaling up. In *Proc. International Conference on Machine Learning*, 1995.

[15] Z. Ghahramani and M. I. Jordan. Factorial hidden Markov models. *Machine Learning*, 1997.

[16] D. Koller and R. Parr. Computing factored value functions for policies in structured MDPs. In *Proc. IJCAI'99*, 1999.

[17] A. Rodriguez, R. Parr, and D. Koller. Reinforcement learning using approximate belief states. In S. A. Solla, T. K. Leen, and K.-R. Müller, editors, *Advances in Neural Information Processing Systems*, volume 12. The MIT Press, Cambridge, 2000.

[18] L. Chrisman. Reinforcement learning with perceptual aliasing: The perceptual distinctions approach. In *Tenth National Conference on AI*, 1992.
